# Spherical Units as Dynamic Consequential Regions: Implications for Attention, Competition and Categorization

Stephen José Hanson*
Learning and Knowledge
Acquisition Group
Siemens Corporate Research
Princeton, NJ 08540

Mark A. Gluck
Center for Molecular &
Behavioral Neuroscience
Rutgers University
Newark, NJ 07102

## Abstract

Spherical Units can be used to construct dynamic reconfigurable consequential regions, the geometric bases for Shepard's (1987) theory of stimulus generalization in animals and humans. We derive from Shepard's (1987) generalization theory a particular multi-layer network with dynamic (centers and radii) spherical regions which possesses a specific mass function (Cauchy). This learning model generalizes the configural-cue network model (Gluck & Bower 1988): (1) configural cues can be learned and do not require pre-wiring the power-set of cues, (2) Consequential regions are continuous rather than discrete and (3) Competition amoungst receptive fields is shown to be increased by the global extent of a particular mass function (Cauchy). We compare other common mass functions (Gaussian; used in models of Moody & Darken; 1989, Krushke, 1990) or just standard backpropogation networks with hyperplane/logistic hidden units showing that neither fare as well as models of human generalization and learning.

## 1 The Generalization Problem

Given a favorable or unfavorable consequence, what should an organism assume about the contingent stimuli? If a moving shadow overhead appears prior to a hawk attack what should an organism assume about other moving shadows, their shapes and positions? If a dense food patch is occasioned by a particular density of certain kinds of shrubbery what should the organism assume about other shurbbery, vegetation or its spatial density? In an pattern recognition context, given a character of a certain shape, orientation, noise level etc.. has been recognized correctly what should the system assume about other shapes, orientations, noise levels it has yet to encounter?

Many "generalization" theories assume stimulus similarity represents a "failure to discriminate", rather than a cognitive decision about what to assume is consequential about the stimulus event. In this paper we implement a generalization theory with multilayer architecture and localized kernel functions (cf. Cooper, 1962; Albus 1975; Kanerva, 1984; Hanson & Burr, 1987,1990; Niranjan & Fallside, 1988; Moody & Darken, 1989; Nowlan, 1990; Krushke, 1990) in which the learning system constructs hypotheses about novel stimulus events.

## 2    Shepard's (1987) Generalization Theory

Considerable empirical evidence indicates that when stimuli are represented within an multi-dimensional psychological space, similarity, as measured by stimulus generalization, drops off in an approximate exponential decay fashion with psychological distance (Shepard, 1957, 1987). In comparison to a linear function, a similarity-distance relationship with upwards concave curvature, such as an exponential-decay curve, exaggerates the similarity of items which are nearby in psychological space and minimizes the impact of items which are further away.

Recently, Roger Shepard (1987) has proposed a "Universal Law of Generalization" for stimulus generalization which derives this exponential decay similarity-distance function as a "rational" strategy given minimal information about the stimulus domain (see also Shepard & Kannappan, this volume). To derive the exponential-decay similarity-distance rule, Shepard (1987) begins by assuming that stimuli can be placed within a psychological space such that the response learned to any one stimulus will generalize to another according to an invariant monotonic function of the distance between them. If a stimulus, $O$, is known to have an important consequence, what is the probability that a novel test stimulus, $X$, will lead to the same consequence? Shepard shows, through arguments based on probabilistic reasoning that regardless of the *a priori* expectations for regions of different sizes, this expectation will almost always yield an approximate exponentially decaying gradient away from a central memory point. In particular, very simple geometric constraints can lead to the exponential generalization gradient. Shepard (1987) assumes (1) that the consequential region overlaps the consequential stimulus event. and (2) bounded center symmetric consequential regions of unknown shape and size In the 1-dimensional case it can be shown that g(x) is robust over a wide variety of assumptions for the distribution of p(s); although for p(s) exactly the Erlangian or discrete Gamma, g(x) is exactly Exponential.

We now investigate possible ways to implement a model which can learn consequential regions and appropriate generalization behavior (cf. Shepard, 1990).

## 3    Gluck & Bower's Configural-cue Network Model

The first point of contact is to discrete model due to Gluck and Bower: The configural-cue network model (Gluck & Bower, 1988) The network model adapts its weights (associations) according to Rescorla and Wagner's (1972) model of classical conditioning which is a special case of Widrow & Hoff's (1961) Least-Mean-Squares (LMS) algorithm for training one-layer networks. Presentation of a stimulus pattern is

represented by activating nodes on the input layer which correspond to the pattern's elementary features and pair-wise conjunctions of features.

The configural-cue network model implicitly embodies an exponential generalization (similarity) gradient (Gluck, 1991) as an emergent property of it's stimulus representation scheme. This equivalence can be seen by computing how the number of overlapping active input nodes (similarity) changes as a function of the number of overlapping component cues (distance). If a stimulus pattern is associated with some outcome, the configural-cue model will generalize this association to other stimulus patterns in proportion to the number of common input nodes they both activate.

Although the configural cue model has been successful with various categorization data, there are several limitations of the configural cue model: (1) it is discrete and can not deal adequately with continuous stimuli (2) it possesses a non-adaptable internal representation (3) it can involve the pre-wiring the power set of possible cues Nonetheless, there are several properties that make the Configural Cue model successful that are important to retain for generalizations of this model: (a) the competitive stimulus properties deriving from the delta rule (b) the exponential stimulus generalization property deriving from the successive combinations of higher-order features encoded by hidden units.

## 4  A Continuous Version of Shepard's Theory

We derive in this section a new model which generalizes the configural cue model and derives directly from Shepard's generalization theory. In Figure 1, is shown a one dimensional depiction of the present theory. Similar to Shepard we assume there is a consequential

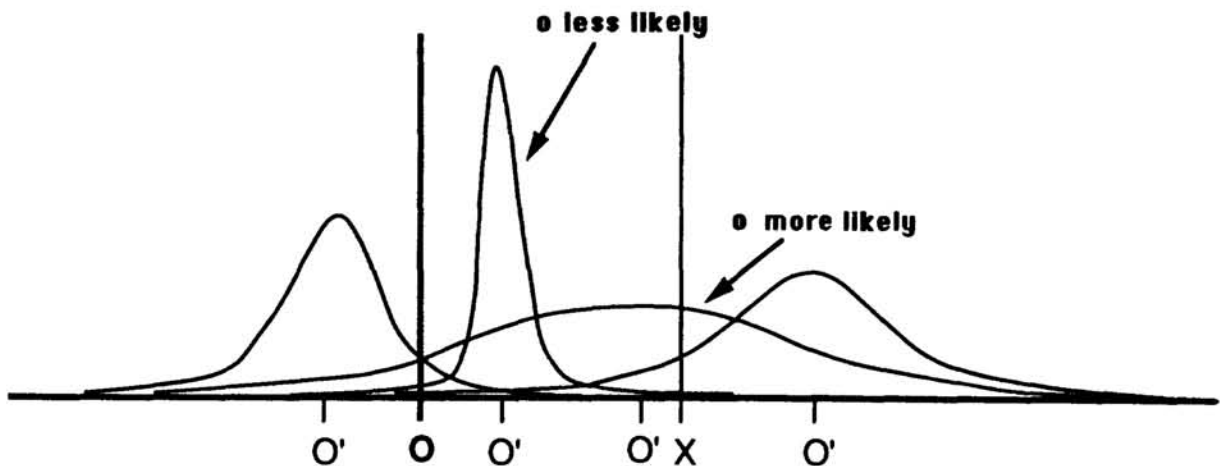

Figure 1: Hypothesis Distributions based on Consequential Region

region associated with a significant stimulus event, **O**. Also similar to Shepard we assume the learning system knows that the significant stimulus event is contained in the consequential region, but does not know the size or location of the consequential region. In absence of this information the learning system constructs hypothesis distributions

(O') which may or maynot be contained in the consequential region but at least overlap the significant stimulus event with some finite probablity measure.  In some hypothesis distributions the significant stimulus event is "typical" in the consequential region, in other hypothesis distributions the significant stimulus event is "rare".  Consequently, the present model differs from Shepard's approach in that the learning system uses the consequential region to project into a continuous hypothesis space in order to construct the conditional probability of the novel stimulus, X, given the significant stimulus event O.

Given no further information on the location and size of the consequential region the learning system averages over all possible locations (equally weighted) and all possible (equally weighted) variances over the known stimulus dimension:

$$g(x) = \int_{s1}^{s2} p(s) \int_{c1}^{c2} p(c) H(x,s,c) \, dc \, ds \qquad (1)$$

In order to derive particular gradients we must assume particular forms for the hypothesis distribution, H(x,s,c).  Although we have investigated many different hypothesis distributions and wieghting functions (p(c), p(s)), we only have space here to report on two bounding cases, one with very "light tails", the Gaussian, and one with very "heavy tails", the Cauchy (see Figure 2).  These two distributions are extremes and provide a test of the robustness of the generalization gradient. At the same time they represent different commitments to the amount of overlap of hidden unit receptive fields and the consequent amount of stimulus competition during learning.

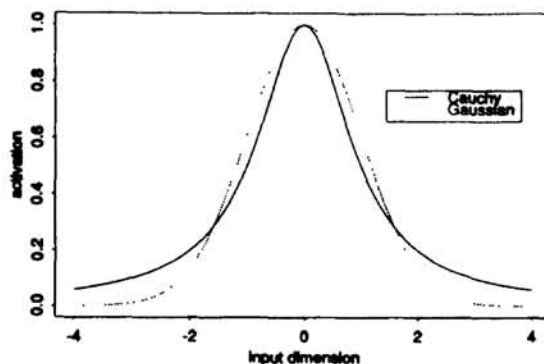

Figure 2: Gaussian compared to the Cauchy: Note heavier Cauchy tail

Equation 2 was numerically integrated (using mathematica), over a large range of variances and a large range of locations using uniform densities representing the weighting functions and both Gaussian and Cauchy distributions representing the hypothesis distributions.  Shown in Figure 3 are the results of the integrations for both the Cauchy and Gaussian distributions.  The resultant gradients are shown by open circles (Cauchy) or stars (Gaussian) while the solid lines show the best fitting exponential gradient.  We note that they approximate the derived gradients rather closely in spite of the fact the underlying forms are quite complex, for example the curve shown for the Cauchy integration is actually:

$$-5 Arctan\left(\frac{x-c\,2}{5}\right)+0.01\left[Arctan\left(100(x-c\,1)\right)\right]+5 Arctan\left(\frac{x+c\,1}{5}\right)- \qquad (2)$$

$$0.01\left[Arctan\left(100(x+c\,2)\right)\right]-\tfrac{1}{2}((c\,2+x)\log(1-s\,1x+x\,2)+(c\,1-x)\log(s\,2-s\,1x+x\,2))-$$

$$\tfrac{1}{2}(c\,1-x)\log(1+s\,1x+x\,2)+(c\,2+x)\log(s\,2+s\,1x+x\,2))$$

Consequently we confirm Shepard's original observation for a continuous version[1] of his theory that the exponential gradient is a robust consequence of minimum information set of assumptions about generalization to novel stimuli.

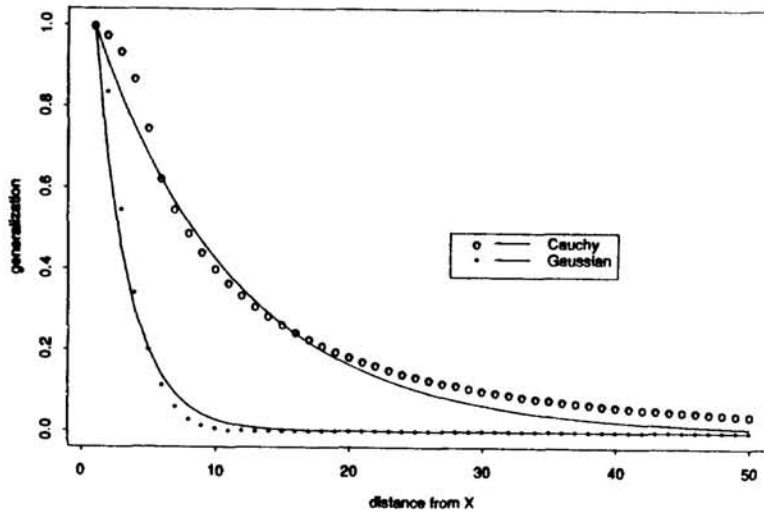

Figure 3: Generalization Gradients Compared to Exponential (Solid Lines)

## 4.1   Cauchy vs Gaussian

As pointed out before the Cauchy has heavier tails than the Gaussian and thus provides more global support in the feature space. This leads to two main differences in the hypothesis distributions:

(1) Global vs Local support: Unlike back-propagation with hyperplanes, Cauchy can be local in the feature space and unlike the Gaussian can have more global effect.

(2) Competition not Dimensional scaling: Dimensional "Attention" in CC and Cauchy multilayer network model is based on competition and effective allocation of resources during learning rather than dimensional contraction or expansion.

Since the stimulus generalization properites of both hypothesis distributions are indistinguishable (both close to exponential) it is important to compare categorization results based on a multilayer gradient descent model using both the Cauchy and Gaussian as hidden node functions.

## 5   Comparisons with Human Categorization Performance

We consider in the final section two experiments from human learning literature which constrain categorization results. The model was a multilayer network using standard gradient descent in the radius, location and second layer weights of either Cauchy or Gaussian functions in hidden units.

### 5.1   Shepard, Hovland and Jenkins (1961)

In order to investigate adults ability to learn simple classification SH&J used eight 3-dimensional stimuli (corners of the cube) representing seperable stimuli like shape, color or size. Of the 70 possible 4-exempler dichotomies there are only six unique 4 exemplar dichotomies which ignor the specific stimulus dimension.

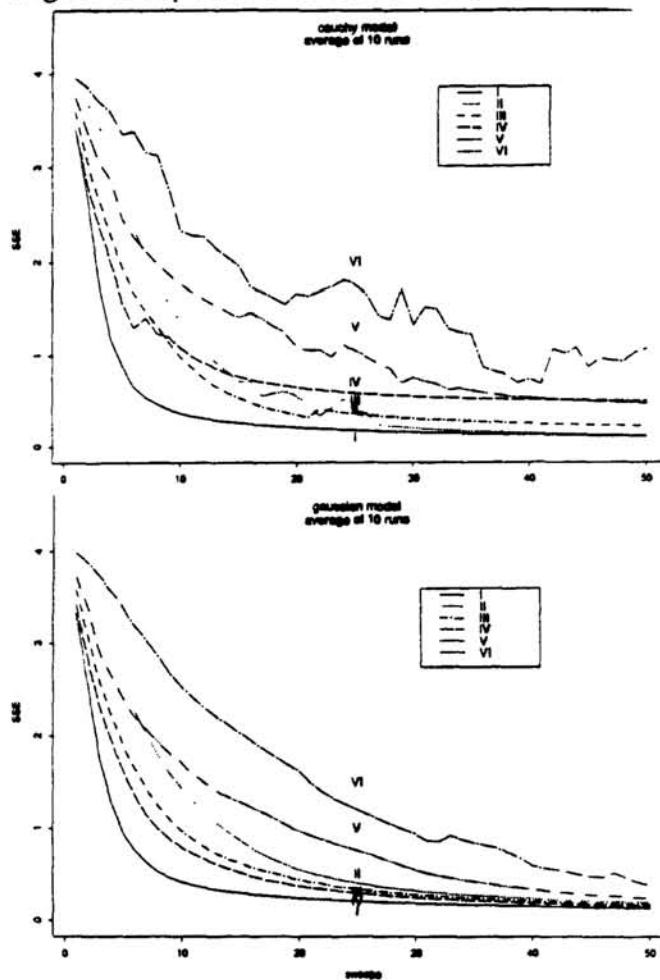

Figure 4: Classification Learning Rate for Gaussian and Cauchy on SHJ stimuli

These dichotomies involve both linearly separable and nonlinearly separable classifications as well as selective dependence on a specific dimension or dimensions.

For both measures of trials to learn and the number of errors made during learning the order of difficulty was (easiest) I<II<III<IV<V<VI (hardest).

In Figure 4, both the Cauchy model and the Gaussian model was compared using the SHJ stimuli. Note that the Gaussian model misorders the 6 classification tasks: I<IV<III<II<V<VI while the Cauchy model conforms with the human performance.

## 5.2    Medin and Schwanenflugel (1981)

Data suitable to illustrate the implications of this non-linear stimulus generalization gradient for classification learning, are provided by Medin and Schwanenflugel (1981). They contrasted performance of groups of subjects learning pairs of classification tasks, one of which was *linearly separable* and one of which was not. One of the classifications is linearly separable (LS) and the other is not (NLS).

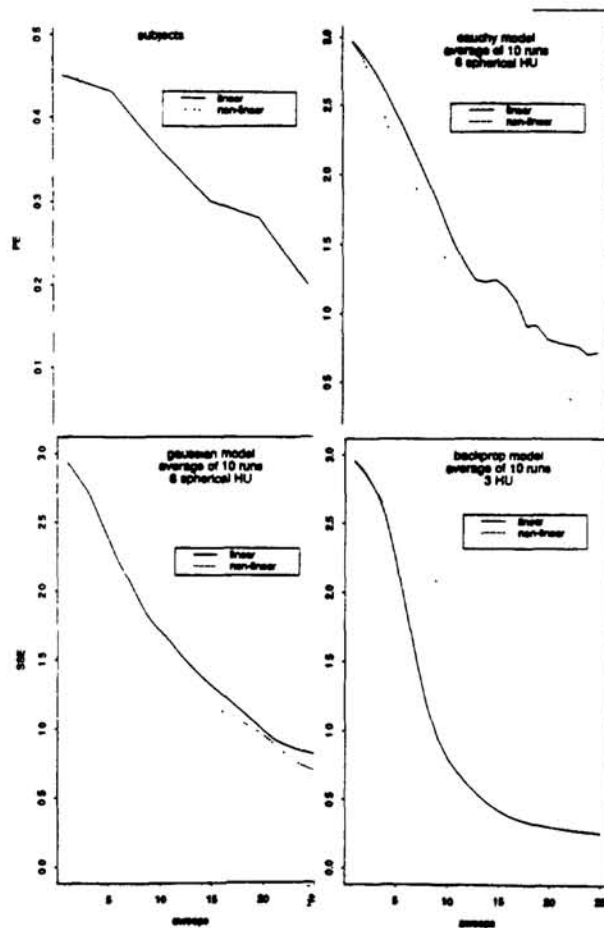

Figure 5: Subjects (a) Cauchy (b) Gaussian (c) and Backprop
(d) Learning performance on the M&S stimuli.

An important difference between the tasks lies in how the between-category and within-category distances are distributed. The linearly separable task is composed of many

"close" (Hamming distance=1) and some "far" (Hamming distance=3) relations, while the non-separable task has a broader distribution of "close", "medium", and "far" between-category distances. These unequal distributions have important implications for models which use a non-linear mapping from distance to similarity. Medin and Schwanenflugel reported reliable and complete results with a four-dimensional task that embodied the same controls for linear separability and inter-exemplar similarities. To evaluate the relative difficulty of the two tasks, Medin & Schwanenflugel compared the average learning curves of subjects trained on these stimuli. Subjects found the linearly separable task (LS) more difficult than the non-linearly separable task (NLS), as indicated by the reduced percentage of errors for the NLS task at all points during training (see next Figure 5--Subjects, top left) In Figure 5 is shown 10 runs of the Cauchy model (top right) note that it, similar to the human performance, had more difficulty with the LS than the NLS separable task. Below this frame is the results for the Gaussian model (bottom left) which does show a slight advantage of learning the NLS task over the LS task. While in the final frame (bottom right) of this series standard backprop actually reverses the speed of learning of each task relative to human performance.

## 6  Conclusions

A continuous version of Shepard's (1987) generalization theory was derived providing for a specific Mass/Activation function (Cauchy) and receptive field distribution. The Cauchy activation function is shown to account for a range of human learning performance while another Mass/Activation function (Gaussian) does not. The present model also generalizes the Configural Cue model to continuous, dynamic, internal representation.

Attention like effects are obtained through competition of Cauchy units as a fixed resource rather than dimensional "shrinking" or "expansion" as in an explicit rescaling of each axes.

Cauchy units are a compromise; providing more global support in approximation than gaussian units and more local support than the hyperplane/logistic units in backpropagation models.

## Footnotes

* Also a member of Cognitive Science Laboratory, Princeton University, Princeton, NJ 08544

[1] N-Dimensional Versions: we generalize the above continuous 1-d model to an N-dimensional model by assuming that a network of Cauchy units can be used to construct a set of consequential regions each possibly composed of several Cauchy receptive fields. Consequently, dimensions can be differentially weighted by subsets of cauchy units acting in concert could produce metrics like L-1 norms in separable (e.g., shape, size of arbitrary forms) dimension cases while equally weighting dimensions similar to metrics like L-2 norms in integral (e.g., lightness, hue in color) dimension cases.

### References

Albus, J. S. (1975) A new approach to manipulator control: The cerebellar model articulation controller (CMAC), American Society of Engineers, Transactions G (Journal of Dynamic Systems, Measurement and Control) 97(3):220-27.

Cooper, P. (1962) The hypersphere in pattern recognition. Information and Control, 5, 324-346.

M. A. Gluck (1991) Stimulus generalization and representation in adaptive network models of category learning. Psychological Science, 2, 1, 1-6.

M. A. Gluck & G. H. Bower, (1988), Evaluating an adaptive network model of human learning. Journal of Memory and Language, 27, 166-195.

Hanson, S. J. and Burr, D. J. (1987) Knowledge Representation in Connectionist Networks, Bellcore Technical Report.

Hanson, S. J. and Burr, D. J. (1990) What Connectionist models learn: Learning and Representation in Neural Networks. Behavioral and Brain Sciences.

Kanerva, P. (1984) Self propagating search : A unified theory of memory; Ph.D. Thesis, Stanford University.

Kruschke, J. (1990) A connectionist model of category learning, Ph.D. Thesis, UC Berkeley.

Medin D. L., & Schanwenflugel, P. J. (1981) Linear seperability in classification learning. Journal of Experimental Psychology: Human Learning and Memory, 7, 355-368.

Moody .J. and Darken, C (1989) Fast learning in networks of locally-tuned processing units, Neural Computation, 1, 2, 281-294.

Niranjan M. & Fallside, F. (1988) Neural networks and radial basis functions in classifying static speech patterns, Technical Report, CUED/FINFENG TR22, Cambridge University.

Nowlan, S. (1990) Max Likelihood Competition in RBF Networks. Technical Report CRG-TR-90-2, University of Toronto.

R. A. Rescorla A. R. Wagner (1972) A theory of Pavlovian conditioning: Variations in the effectiveness of reinforcement and non-reinforcement. A. H. Black W. F. Prokasy (Eds.) Classical conditioning II: Current research and theory, 64-99 Appleton-Century-Crofts: New York.

R. N. Shepard (1958), Stimulus and response generalization: Deduction of the generalization gradient from a trace model, Psychological Review 65, 242-256

Shepard, R. N. (1987) Toward a Universal Law of Generalization for Psychological Science. Science, 237.

R. N. Shepard, C. I. Hovland & H. M. Jenkins (1961), Learning and memorization of classifications, Psychological Monographs, 75, 1-42

A B. Widrow & M. E. Hoff (1960) Adaptive switching circuits, Institute of Radio Engineers, Western Electronic Show and Convention, Convention Record, 4, 96-194
